# A Network of Localized Linear Discriminants

**Martin S. Glassman**
Siemens Corporate Research
755 College Road East
Princeton, NJ 08540
msg@siemens.siemens.com

## Abstract

The localized linear discriminant network (LLDN) has been designed to address classification problems containing relatively closely spaced data from different classes (*encounter zones* [1], the accuracy problem [2]). Locally trained hyperplane segments are an effective way to define the decision boundaries for these regions [3]. The LLD uses a modified perceptron training algorithm for effective discovery of separating hyperplane/sigmoid units within narrow boundaries. The basic unit of the network is the *discriminant receptive field* (DRF) which combines the LLD function with Gaussians representing the dispersion of the local training data with respect to the hyperplane. The DRF implements a local distance measure [4], and obtains the benefits of networks of localized units [5]. A constructive algorithm for the two-class case is described which incorporates DRF's into the hidden layer to solve local discrimination problems. The output unit produces a smoothed, piecewise linear decision boundary. Preliminary results indicate the ability of the LLDN to efficiently achieve separation when boundaries are narrow and complex, in cases where both the "standard" multilayer perceptron (MLP) and k-nearest neighbor (KNN) yield high error rates on training data.

## 1 The LLD Training Algorithm and DRF Generation

The LLD is defined by the hyperplane normal vector $V$ and its "midpoint" $M$ (a translated origin [1] near the center of gravity of the training data in feature space). Incremental corrections to $V$ and $M$ accrue for each training token feature vector $Y_j$ in the training set, as illustrated in figure 1 (exaggerated magnitudes). The surface of the hyperplane is appropriately moved either towards or away from $Y_j$ by rotating $V$, and shifting $M$ along

the axis defined by $V$. $M$ is always shifted towards $Y_j$ in the "radial" direction $R_j$ (which is the component of $D_j$ orthogonal to $V$, where $D_j = Y_j - M$):

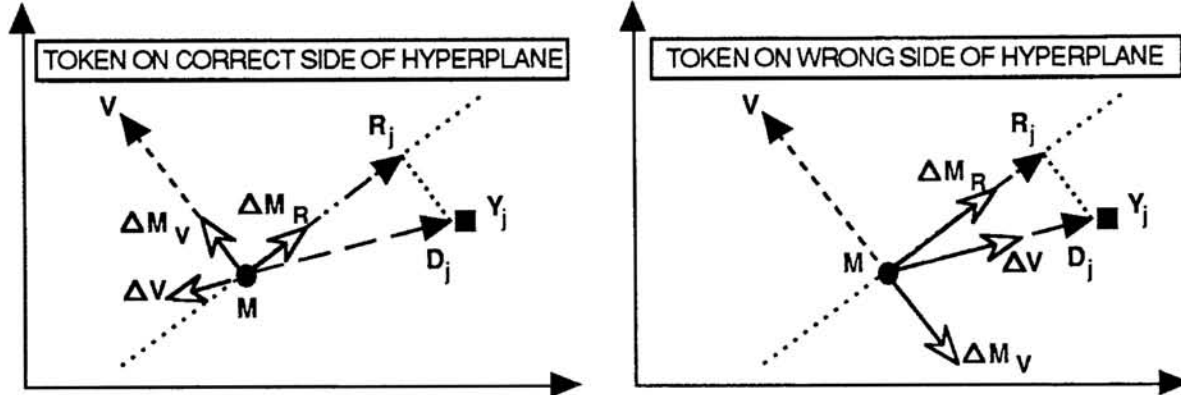

Figure 1: LLD incremental correction vectors associated with training token $Y_j$ are shown above, and the corresponding LLD update rules below:

$$\Delta \vec{V} = \mu(n) \sum_j \Delta \vec{V}_j = \mu(n) \sum_j (\frac{-s_c w_c \varepsilon_j}{\|\vec{D}_j\|}) \vec{D}_j$$

$$\Delta \vec{M}_{\vec{V}} = \gamma(n) \sum_j \Delta \vec{M}_{\vec{V}_j} = \gamma(n) \sum_j (-s_c w_c \varepsilon_j) \vec{V}$$

$$\Delta \vec{M}_{\vec{R}} = \beta(n) \sum_j \Delta \vec{M}_{\vec{R}_j} = \beta(n) \sum_j (w_c \varepsilon_j) \vec{R}_j$$

The batch mode summation is over tokens in the local training set, and $n$ is the iteration index. The polarity of $\Delta V_j$ and $\Delta M_{R_j}$ is set by $s_c$ ($c$ = the class of $Y_j$), where $s_c = 1$ if $Y_j$ is classified correctly, and $s_c = -1$ if not. Corrections for each token are scaled by a sigmoidal error term: $\varepsilon_j = 1/(1 + \exp((s_c \eta/\lambda) \mid \vec{V}^T \vec{D}_j \mid))$, a function of the distance of the token to the plane, the sign of $s_c$, and a data-dependent scaling parameter: $\lambda = \mid \vec{V}^T [\vec{B}_0 - \vec{B}_1] \mid$, where $\eta$ is a fixed (experimental) scaling parameter. The scaling of the sigmoid is proportional to an estimate of the boundary region width along the axis of $V$. $B_c$ is a weighted average of the class $c$ token vectors: $\vec{B}_c(n+1) = (1 - \alpha)\vec{B}_c(n) + \alpha w_c \sum_{j \in c} \varepsilon_{j,c}(n)\vec{Y}_j(n)$, where $\epsilon_{j,c}$ is a sigmoid with the same scaling as $\varepsilon_j$, except that it is centered on $B_c$ instead of $M$, emphasizing tokens of class $c$ nearest the hyperplane surface. For small $\eta$'s, $B_c$ will settle near the cluster center of gravity, and for large $\eta$'s, $B_c$ will approach the tokens closest to the hyperplane surface. (The rate of the movement of $B_c$ is limited by the value of $\alpha$, which is not critical.) The inverse of the number of tokens in class $c$, $w_c$, balances the weight of the corrections from each class. If a more Bayesian-like solution is required, the slope of $\varepsilon$ can be made class dependent (for example, replacing $\eta$ with $\eta_c \propto w_c$). Since the slope of the sigmoid error term is limited and distribution dependent, the use of $w_c$, along with the nonlinear weighting of tokens near the hyperplane surface, is important for the development of separating planes in relatively narrow boundaries (the assumption is that the distributions near these boundaries are non-Gaussian). The setting of $\eta$ simultaneously (for convenience) controls the focus on the "inner edges" of the class clusters and the slope of the sigmoid relative to the distance between the inner edges, with some resultant control over generalization performance. This local scaling of the error also aids the convergence rate. The range of good values for $\eta$ has been found to be reasonably wide, and identical

values have been used successfully with speech, ecg, and synthetic data; it could also be set/optimized using cross-validation. Separate adaptive learning rates ($\mu(n)$, $\gamma(n)$, and $\beta(n)$) are used in order to take advantage of the distinct nature of the geometric function of each component. Convergence is also improved by maintaining $M$ within the local region; this controls the rate at which the hyperplane can sweep through the boundary region, making the effect of $\Delta V$ more predictable. The LLD normal vector update is simply: $\vec{V}(n+1) = (\vec{V}(n) + \Delta\vec{V})/\|\vec{V}(n) + \Delta\vec{V}\|$, so that $V$ is always normalized to unit magnitude. The midpoint is just shifted: $\vec{M}(n+1) = \vec{M}(n) + \Delta\vec{M}_{\vec{R}} + \Delta\vec{M}_{\vec{V}}$ .

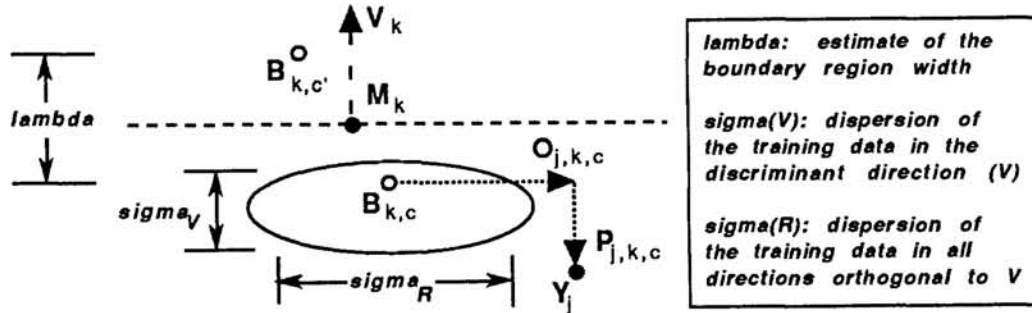

Figure 2: Vectors and parameters associated with the DRF for class $c$, for LLD k

DRF's are used to localize the response of the LLD to the region of feature space in which it was trained, and are constructed after completion of LLD training. Each DRF represents one class, and the localizing component of the DRF is a Gaussian function based on simple statistics of the training data for that class. Two measures of the dispersion of the data are used: $\sigma_V$ ("normal" dispersion), obtained using the mean average deviation of the lengths of $P_{j,k,c}$, and $\sigma_R$ ("radial" dispersion), obtained correspondingly using the $O_{j,k,c}$'s. (As shown, $P_{j,k,c}$ is the normal component, and $O_{j,k,c}$ the radial component of $Y_j - B_{k,c}$.) The output in response to an input vector $Y_j$ from the class $c$ DRF associated with the LLD $k$ is $\phi_{j,k,c}$:

$$\phi_{j,k,c} = \Theta_{k,c}(\varepsilon_{j,k}-0.5)/\exp(\sqrt{d^2_{\vec{V}j,k,c} + d^2_{\vec{R}j,k,c}}); \quad \varepsilon_{j,k} = 1/(1+\exp((\eta/\lambda_k)\,|\,\vec{V}_k^T[\vec{Y}_j-\vec{M}_k]\,|))$$

Two components of the DRF incorporate the LLD discriminant; one is the sigmoid error function used in training the LLD but shifted down to a value of zero at the hyperplane surface. The other is $\Theta_{k,c}$, which is 1 if $Y_j$ is on the class $c$ side of LLD $k$, and zero if not. (In retrospect, for generalization performance, it may not be desirable to introduce this discontinuity to the discriminant component.) The contribution of the Gaussian is based on the normal and radial dispersion weighted distances of the input vector to $B_{k,c}$:

$$d_{\vec{V}j,k,c} = \|\vec{P}_{j,k,c}\|/\sigma_{\vec{V},k,c}, \quad and. \quad d_{\vec{R}j,k,c} = \|\vec{O}_{j,k,c}\|/\sigma_{\vec{R},k,c}.$$

## 2   Network Construction

Segmentation of the boundary between classes is accomplished by "growing" LLD's within the boundary region. An LLD is initialized using a closely spaced pair of tokens from each class. The LLD is grown by adding nearby tokens to the training set, using the k-nearest neighbors to the LLD midpoint at each growth stage as candidates for permanent inclusion. Candidate DRF's are generated after incremental training of the LLD to accommodate each

new candidate token. Two error measures are used to assess the effect of each candidate, the peak value of $\varepsilon_j$ over the local training set, and $\varpi$, which is a measure of misclassification error due to the receptive fields of the candidate DRF's extending over the *entire* training set. The candidate token with the lowest average $\varpi$ is permanently added, as long as both its $\varepsilon_j$ and $\varpi$ are below fixed thresholds. Growth the the LLD is halted if no candidate has both error measures below threshold. The $\varepsilon_j$ and $\varpi$ thresholds directly affect the granularity of the DRF representation of the data; they need to be set to minimize the number of DRF's generated, while allowing sufficient resolution of local discrimination problems. They should perhaps be adaptive so as to encourage coarse grained solutions to develop before fine grain structure.

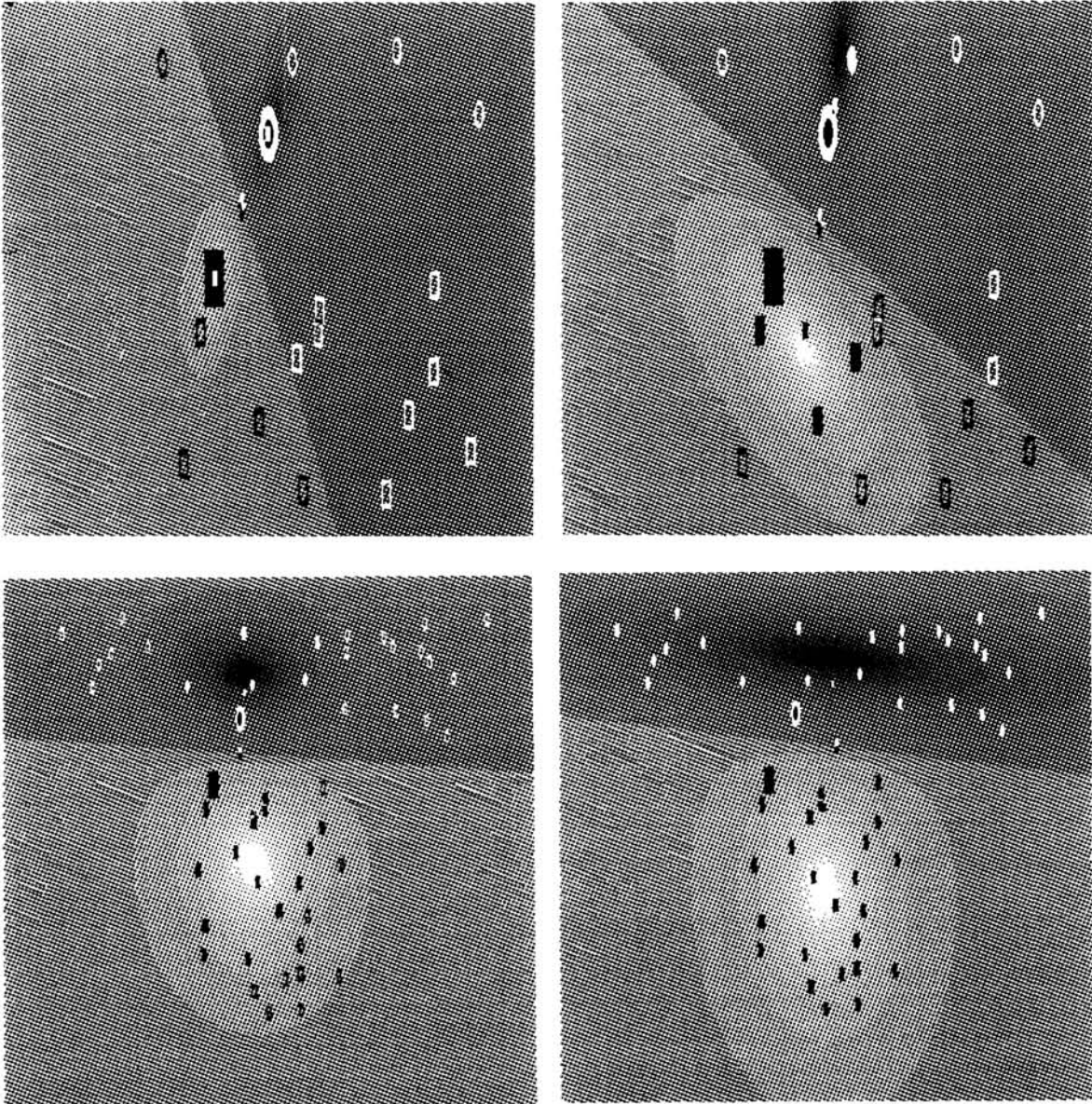

Figure 3: Four "snapshots" in the growth of an LLD/DRF pair. The upper two are "close-ups." The initial LLD/DRF pair is shown in the upper left, along with the seed pair. Filled rectangles and ellipses represent the tokens from each class in the permanent local training set at each stage. The large markers are the B points, and the cross is the LLD midpoint. The amplitude of the DRF outputs are coded in greyscale.

At this point the DRF's are fixed and added to the network; this represents the addition of two new localized features available for use by the network's output layer in solving the global discrimination problem. In this implementation, the output "layer" is a single LLD used to generate a two-class decision. The architecture is shown below:

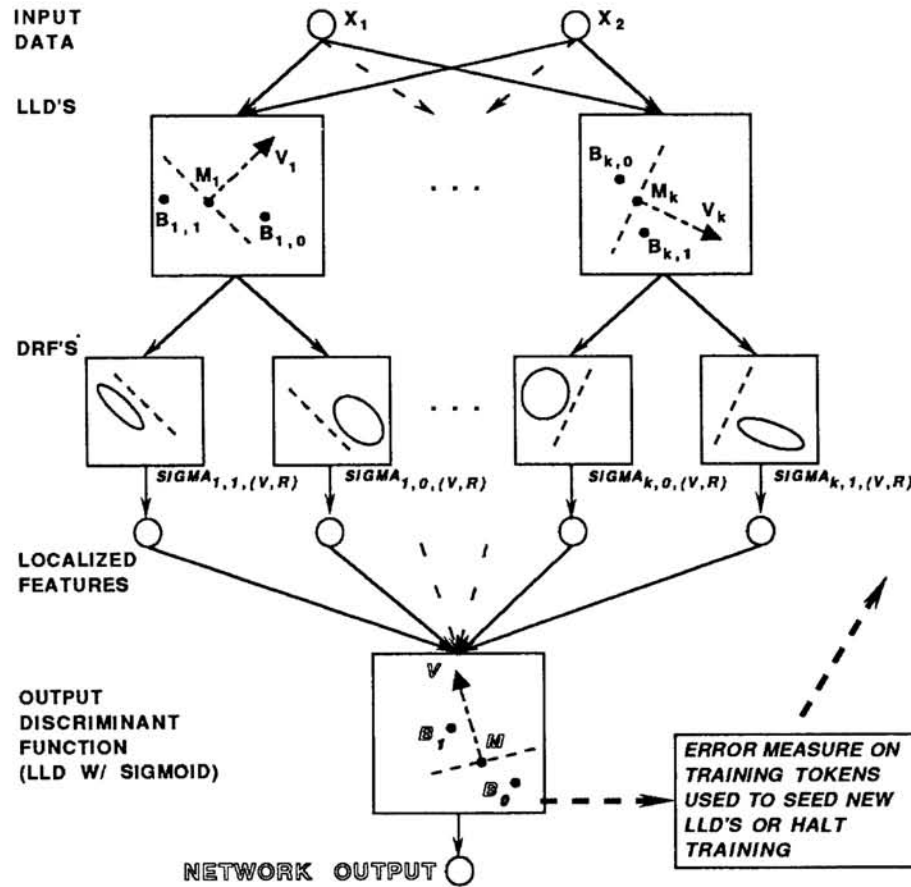

Figure 4: LLDN architecture for a two-dimensional, two-class problem

The ouput unit is completely retrained after addition of a new DRF pair, using the *entire* training set. The output of the network to the input $Y_j$ is: $\varphi_j = 1/(1+\exp((\eta/\lambda_O)\vec{\mathcal{V}}^T[\vec{\Phi}_j - \mathcal{M}]))$, where $\lambda_O = |\vec{\mathcal{V}}^T[\vec{\mathcal{B}}_0 - \vec{\mathcal{B}}_1]|$, and $\vec{\Phi}_j = [\phi_{j,1}, \ldots, \phi_{j,p}]$ is the $p$ dimensional vector of DRF outputs presented to the output unit. $\mathcal{V}$ is the output LLD normal vector, $\mathcal{M}$ the midpoint, and $\mathcal{B}_c$'s the cluster edge points in the internal feature space. The output error for each token is then used to select a new seed pair for development of the next LLD/DRF pair. If all tokens are classified with sufficient confidence, of course, construction of the LLDN is complete. There are three possibilities for insufficient confidence: a token is covered by a DRF of the wrong class, it is not yet covered sufficiently by any DRF's, or it is in a region of "conflict" between DRF's of different classes. A heuristic is used to prevent the repeated selection of the same seed pair tokens, since there is no guarantee that a given DRF will significantly reduce the error for the data it covers after output unit retraining. This heuristic alternates between the types of error and the class for selection of the primary seed token. Redundancy in DRF shapes is also minimized by error-weighting the dispersion computations so that the resultant Gaussian focuses more on the higher error regions of the local training data. A simple but reasonably effective pruning algorithm was incorporated to further eliminate unnecessary DRF's.

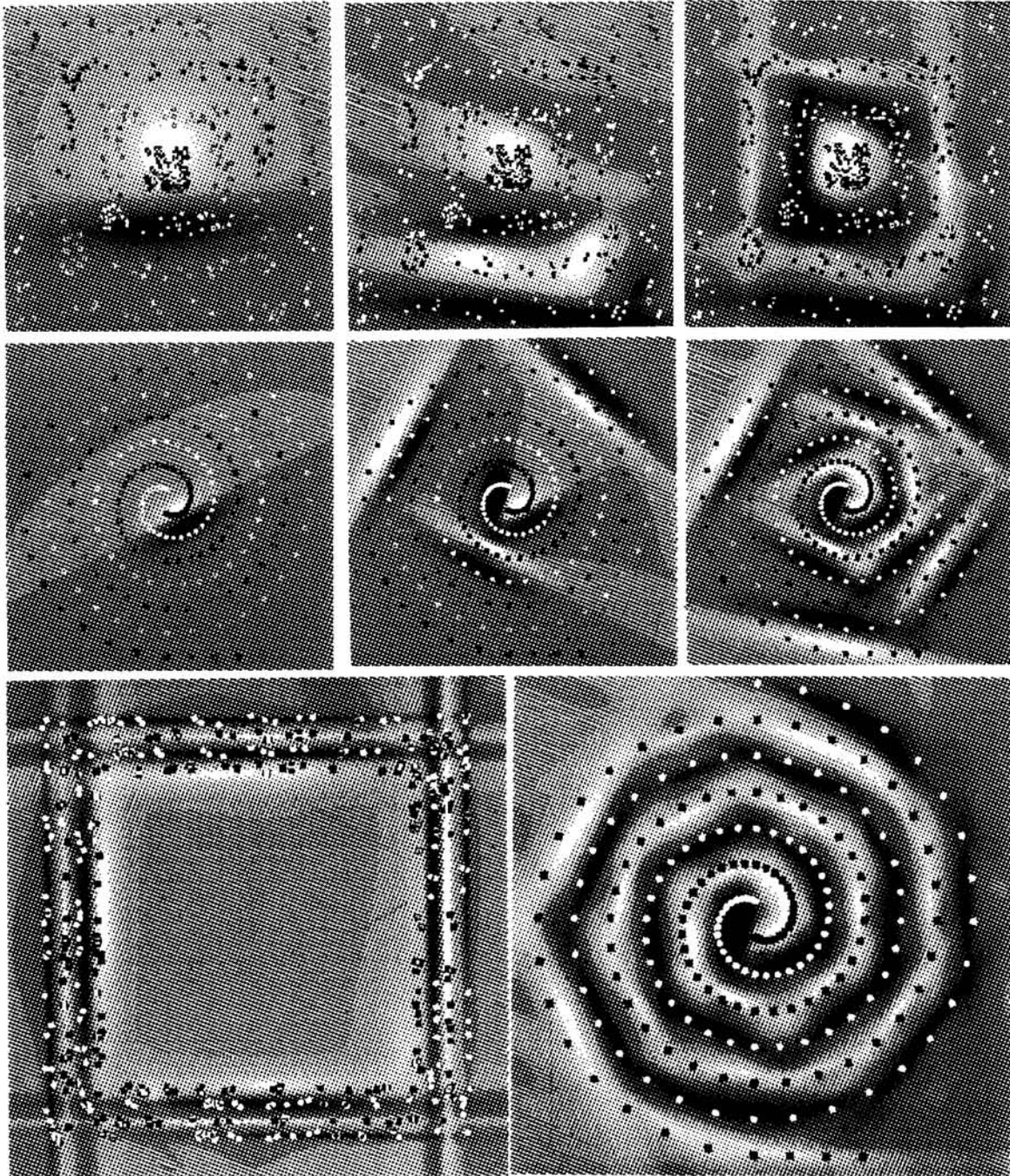

Figure 5: Network response plots illustrating network development. The upper two sequences, beginning with the first LLD/DRF pair, and the bottom two plots show final network responses for these two problems. A solution to a harder version of the nested squares problem is on the lower left.

## 3   Experimental Results

The first experiment demonstrates comparative convergence properties of the LLD and a single hyperplane trained by the standard generalized delta rule (GDR) method (no hidden units, single output unit "network" is used) on 14 linearly separable, minimal consonant

pair data sets. The data is 256 dimensional (time/frequency matrix, described in [6]), with 80 exemplars per consonant. The results compare the best performance obtainable from each technique. The LLD converges roughly 12 times faster in iteration counts. The GDR often fails to .completely separate f/th, f/v, and s/sh; in the results in figure 6 it fails on the f/th data set at a plateau of 25% error. In both experiments described in this paper, networks were run for relatively long times to insure confidence in declaring failure to

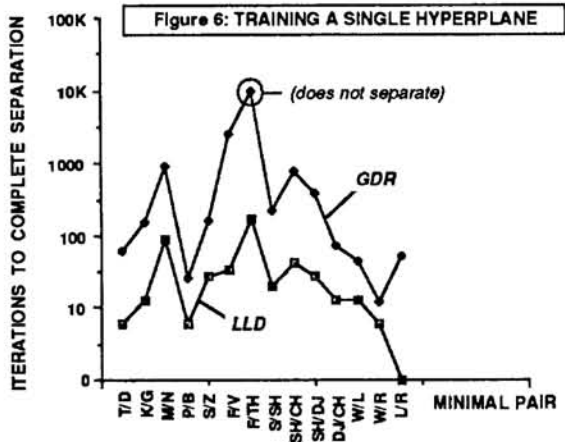

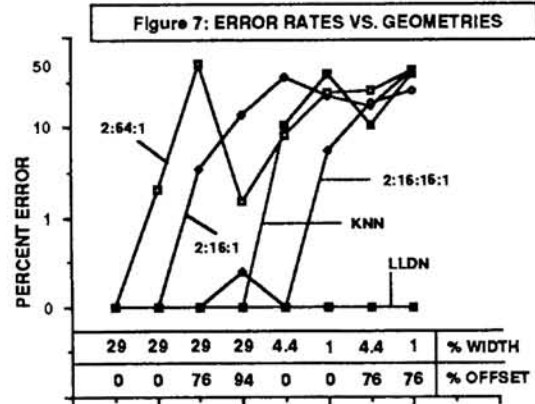

solve the problem. The second experiment involves complete networks on synthetic two-dimensional problems. Two examples of the nested squares problem (random distributions of tokens near the surface of squares of alternating class, 400 tokens total) are shown in figure 5. Two parameters controlling data set generation are explored: the relative boundary region width, and the relative offset from the origin of the data set center of gravity (while keeping the upper right corner of the outside square near the (1,1) coordinate); all data is kept within the unit square (except for geometry number 2). Relative boundary widths of 29%, 4.4%, and 1% are used with offsets of 0%, 76%, and 94%. The best results over parameter settings are reported for each network for each geometry. Four MLP architectures were used: 2:16:1, 2:32:1, 2:64:1, and 2:16:16:1; all of these converge to a solution for the easiest problem (wide boundaries, no offset), but all eventually fail as the boundaries narrow and/or the offset increases. The worst performing net (2:64:1) fails for 7/8 problems (maximum error rate of 49%); the best net (2:16:16:1) fails in 3/8 (maximum of 24% error). The LLDN is 1 to 3 orders of magnitude faster in cpu time when the MLP does converge, even though it does not use adaptive learning rates in this experiment. (The average running time for the LLDN was 34 minutes; for the MLP's it was 3481 minutes [Stardent 3040, single cpu], but which includes non-converging runs. The 2:16:16:1 net did, however, take 4740 minutes to solve problem 6, which was solved in 7 minutes by the LLDN.) The best LLDN's converge to zero errors over the problem set (fig. 6), and are not too sensitive to parameter variation, which primarily affect convergence time and number of DRF's generated. In contrast, finding good values for learning rate and momentum for the MLP's for each problem was a time-consuming process. The effect of random weight initialization in the MLP is not known because of the long running times required. The KNN error rate was estimated using the leave-one-out method, and yields error rates of 0%, 10.5%, and 38.75% (for the best k's) respectively for the three values of boundary width. The LLDN is insensitive to offset and scale (like the KNN) because of the use of the local origin $(M)$ and error scaling $(\lambda)$. While global offset and scaling problems for the MLP can be ameliorated through normalization and origin translation, this method cannot guarantee elimination of local offset and scaling problems. The LLDN's utilization

of DRF's was reasonably efficient, with the smallest networks (after pruning) using 20, 32, and 54 DRF's for the three boundary widths. A simple pruning algorithm, which starts up after convergence, iteratively removes the DRF's with the lowest connection weights to the output unit (which is retrained after each link is removed). A range of roughly 20% to 40% of the DRF's were removed before developing misclassification errors on the training sets. The LLDN was also tested on the "two-spirals" problem, which is know to be difficult for the standard MLP methods. Because of the boundary segmentation process, solution of the two-spirals problem was straightforward for the LLDN, and could be tuned to converge in as fast as 2.5 minutes on an Apollo DN10000. The solution shown in fig. 5 uses 50 DRF's (not pruned). The generalization pattern is relatively "nice" (for training on the sparse version of the data set), and perhaps demonstrates the practical nature of the smoothed piecewise linear boundary for nonlinear problems.

## 4  Discussion

The effect of LLDN parameters on generalization performance needs to be studied. In the nested squares problem it is clear that the MLP's will have better generalization *when they converge*; this illustrates the potential utility of a multi-scale approach to developing localized discriminants. A number of extensions are possible: Localized feature selection can be implemented by simply zeroing components of $V$. The DRF Gaussians could model the radial dispersion of the data more effectively (in greater than two dimensions) by generating principal component axes which are orthogonal to $V$. Extension to the multiclass case can be based on DRF sets developed for discrimination between each class and all other classes, using the DRF's as features for a multi-output classifier. The use of multiple hidden layers offers the prospect of more complex localized receptive fields. Improvement in generalization might be gained by including a procedure for merging neighboring DRF's. While it is felt that the LLD parameters should remain fixed, it may be advantageous to allow adjustment of the DRF Gaussian dispersions as part of the output layer training. A stopping rule for LLD training needs to be developed so that adaptive learning rates can be utilized effectively. This rule may also be useful in identifying poor token candidates early in the incremental LLD training.

References

[1] J. Sklansky and G.N. Wassel. *Pattern Classifiers and Trainable Machines.* Springer Verlag, New York, 1981

[2] S. Makram-Ebeid, J.A. Sirat, and J.R. Viala. A rationalized error backpropagation learning algorithm. *Proc. IJCNN*, 373-380, 1988

[3] J. Sklansky, and Y. Park. Automated design of multiple-class piecewise linear classifiers. *Journal of Classification*, 6:195-222, 1989

[4] R.D. Short, and K. Fukanaga. A new nearest neighbor distance measure. *Proc. Fifth Intl. Conf. on Pattern Rec.*, 81-88

[5] R. Lippmann. A critical overview of neural network pattern classifiers. *Neural Networks for Signal Processing (IEEE)*, 267-275, 1991

[6] M.S. Glassman and M.B. Starkey. Minimal consonant pair discrimination for speech therapy. *Proc. European Conf. on Speech Comm. and Tech.*, 273-276, 1989